# The Concave-Convex Procedure (CCCP)

**A. L. Yuille and Anand Rangarajan** *
Smith-Kettlewell Eye Research Institute,
2318 Fillmore Street,
San Francisco, CA 94115, USA.
Tel. (415) 345-2144. Fax. (415) 345-8455.
Email yuille@ski.org

* Prof. Anand Rangarajan. Dept. of CISE, Univ. of Florida Room 301, CSE Building Gainesville, FL 32611-6120 Phone: (352) 392 1507 Fax: (352) 392 1220 e-mail: anand@cise.ufl.edu

## Abstract

We introduce the Concave-Convex procedure (CCCP) which constructs discrete time iterative dynamical systems which are guaranteed to monotonically decrease global optimization/energy functions. It can be applied to (almost) any optimization problem and many existing algorithms can be interpreted in terms of CCCP. In particular, we prove relationships to some applications of Legendre transform techniques. We then illustrate CCCP by applications to Potts models, linear assignment, EM algorithms, and Generalized Iterative Scaling (GIS). CCCP can be used both as a new way to understand existing optimization algorithms and as a procedure for generating new algorithms.

## 1 Introduction

There is a lot of interest in designing discrete time dynamical systems for inference and learning (see, for example, [10], [3], [7], [13]).

This paper describes a simple geometrical Concave-Convex procedure (CCCP) for constructing discrete time dynamical systems which can be guaranteed to decrease almost any global optimization/energy function (see technical conditions in section (2)).

We prove that there is a relationship between CCCP and optimization techniques based on introducing auxiliary variables using Legendre transforms. We distinguish between *Legendre min-max* and *Legendre minimization*. In the former, see [6], the introduction of auxiliary variables converts the problem to a min-max problem where the goal is to find a saddle point. By contrast, in *Legendre minimization*, see [8], the problem remains a minimization one (and so it becomes easier to analyze

convergence). CCCP relates to *Legendre minimization* only and gives a geometrical perspective which complements the algebraic manipulations presented in [8].

CCCP can be used both as a new way to understand existing optimization algorithms and as a procedure for generating new algorithms. We illustrate this by giving examples from Potts models, EM, linear assignment, and Generalized Iterative Scaling. Recently, CCCP has also been used to construct algorithms to minimize the Bethe/Kikuchi free energy [13].

We introduce CCCP in section (2) and relate it to Legendre transforms in section (3). Then we give examples in section (4).

## 2 The Concave-Convex Procedure (CCCP)

The key results of CCCP are summarized by Theorems 1,2, and 3.

Theorem 1 shows that any function, subject to weak conditions, can be expressed as the sum of a convex and concave part (this decomposition is not unique). This implies that CCCP can be applied to (almost) any optimization problem.

**Theorem 1.** *Let $E(\vec{x})$ be an energy function with bounded Hessian $\partial^2 E(\vec{x})/\partial \vec{x} \partial \vec{x}$. Then we can always decompose it into the sum of a convex function and a concave function.*

Proof. *Select any convex function $F(\vec{x})$ with positive definite Hessian with eigenvalues bounded below by $\epsilon > 0$. Then there exists a positive constant $\lambda$ such that the Hessian of $E(\vec{x}) + \lambda F(\vec{x})$ is positive definite and hence $E(\vec{x}) + \lambda F(\vec{x})$ is convex. Hence we can express $E(\vec{x})$ as the sum of a convex part, $E(\vec{x}) + \lambda F(\vec{x})$, and a concave part $-\lambda F(\vec{x})$.*

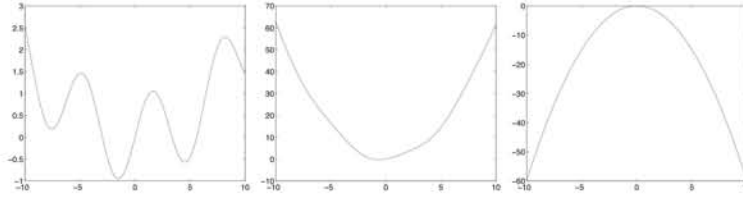

Figure 1: Decomposing a function into convex and concave parts. The original function (Left Panel) can be expressed as the sum of a convex function (Centre Panel) and a concave function (Right Panel). (Figure courtesy of James M. Coughlan).

Our main result is given by Theorem 2 which defines the CCCP procedure and proves that it converges to a minimum or saddle point of the energy.

**Theorem 2.** *Consider an energy function $E(\vec{x})$ (bounded below) of form $E(\vec{x}) = E_{vex}(\vec{x}) + E_{cave}(\vec{x})$ where $E_{vex}(\vec{x}), E_{cave}(\vec{x})$ are convex and concave functions of $\vec{x}$ respectively. Then the discrete iterative CCCP algorithm $\vec{x}^t \mapsto \vec{x}^{t+1}$ given by:*

$$\vec{\nabla} E_{vex}(\vec{x}^{t+1}) = -\vec{\nabla} E_{cave}(\vec{x}^t), \tag{1}$$

*is guaranteed to monotonically decrease the energy $E(\vec{x})$ as a function of time and hence to converge to a minimum or saddle point of $E(\vec{x})$.*

Proof. *The convexity and concavity of $E_{vex}(.)$ and $E_{cave}(.)$ means that $E_{vex}(\vec{x}_2) \geq E_{vex}(\vec{x}_1) + (\vec{x}_2 - \vec{x}_1) \cdot \vec{\nabla} E_{vex}(\vec{x}_1)$ and $E_{cave}(\vec{x}_4) \leq E_{cave}(\vec{x}_3) + (\vec{x}_4 - \vec{x}_3) \cdot \vec{\nabla} E_{cave}(\vec{x}_3)$, for all $\vec{x}_1, \vec{x}_2, \vec{x}_3, \vec{x}_4$. Now set $\vec{x}_1 = \vec{x}^{t+1}, \vec{x}_2 = \vec{x}^t, \vec{x}_3 = \vec{x}^t, \vec{x}_4 = \vec{x}^{t+1}$. Using the algorithm definition (i.e. $\vec{\nabla} E_{vex}(\vec{x}^{t+1}) = -\vec{\nabla} E_{cave}(\vec{x}^t))$ we find that $E_{vex}(\vec{x}^{t+1}) + E_{cave}(\vec{x}^{t+1}) \leq E_{vex}(\vec{x}^t) + E_{cave}(\vec{x}^t)$, which proves the claim.*

We can get a graphical illustration of this algorithm by the reformulation shown in figure (2) (suggested by James M. Coughlan). Think of decomposing the energy function $E(\vec{x})$ into $E_1(\vec{x}) - E_2(\vec{x})$ where both $E_1(\vec{x})$ and $E_2(\vec{x})$ are convex. (This is equivalent to decomposing $E(\vec{x})$ into a a convex term $E_1(\vec{x})$ *plus* a concave term $-E_2(\vec{x})$). The algorithm proceeds by matching points on the two terms which have the same tangents. For an input $\vec{x}_0$ we calculate the gradient $\vec{\nabla} E_2(\vec{x}_0)$ and find the point $\vec{x}_1$ such that $\vec{\nabla} E_1(\vec{x}_1) = \vec{\nabla} E_2(\vec{x}_0)$. We next determine the point $\vec{x}_2$ such that $\vec{\nabla} E_1(\vec{x}_2) = \vec{\nabla} E_2(\vec{x}_1)$, and repeat.

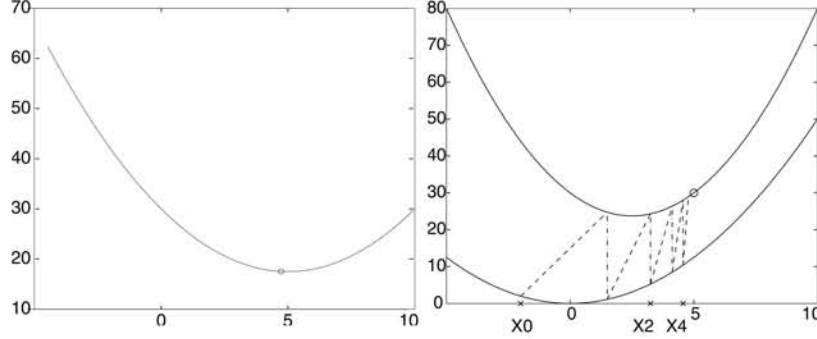

Figure 2: A CCCP algorithm illustrated for Convex minus Convex. We want to minimize the function in the Left Panel. We decompose it (Right Panel) into a convex part (top curve) minus a convex term (bottom curve). The algorithm iterates by matching points on the two curves which have the same tangent vectors, see text for more details. The algorithm rapidly converges to the solution at $x = 5.0$.

We can extend Theorem 2 to allow for linear constraints on the variables $\vec{x}$, for example $\sum_i c_i^\mu x_i = \alpha^\mu$ where the $\{c_i^\mu\}, \{\alpha^\mu\}$ are constants. This follows directly because properties such as convexity and concavity are preserved when linear constraints are imposed. We can change to new coordinates defined on the hyperplane defined by the linear constraints. Then we apply Theorem 1 in this coordinate system.

Observe that Theorem 2 defines the update as an *implicit* function of $\vec{x}^{t+1}$. In many cases, as we will show, it is possible to solve for $\vec{x}^{t+1}$ directly. In other cases we may need an algorithm, or *inner loop*, to determine $\vec{x}^{t+1}$ from $\vec{\nabla} E_{vex}(\vec{x}^{t+1})$. In these cases we will need the following theorem where we re-express CCCP in terms of minimizing a time sequence of *convex update energy functions* $E_{t+1}(\vec{x}^{t+1})$ to obtain the updates $\vec{x}^{t+1}$ (i.e. at the $t^{th}$ iteration of CCCP we need to minimize the energy $E_{t+1}(\vec{x}^{t+1})$). We include linear constraints in Theorem 3.

**Theorem 3**. *Let $E(\vec{x}) = E_{vex}(\vec{x}) + E_{cave}(\vec{x})$ where $\vec{x}$ is required to satisfy the linear constraints $\sum_i c_i^\mu x_i = \alpha^\mu$, where the $\{c_i^\mu\}, \{\alpha^\mu\}$ are constants. Then the update rule for $\vec{x}^{t+1}$ can be formulated as minimizing a time sequence of convex update energy*

*functions $E_{t+1}(\vec{x}^{t+1})$:*

$$E_{t+1}(\vec{x}^{t+1}) = E_{vex}(\vec{x}^{t+1}) + \sum_i x_i^{t+1} \frac{\partial E_{con}}{\partial x_i}(\vec{x}^t) + \sum_\mu \lambda_\mu \{\sum_i c_{i\mu} x_i^{t+1} - \alpha_\mu\}, \quad (2)$$

*where the lagrange parameters $\{\lambda_\mu\}$ impose linear comnstraints.*

Proof. *Direct calculation.*

The convexity of $E_{t+1}(\vec{x}^{t+1})$ implies that there is a unique minimum corresponding to $\vec{x}^{t+1}$. This means that if an inner loop is needed to calculate $\vec{x}^{t+1}$ then we can use standard techniques such as conjugate gradient descent (or even CCCP).

## 3   Legendre Transformations

The Legendre transform can be used to reformulate optimization problems by introducing auxiliary variables [6]. The idea is that some of the formulations may be more effective (and computationally cheaper) than others. We will concentrate on *Legendre minimization*, see [7] and [8], instead of *Legendre min-max* emphasized in [6]. An advantage of *Legendre minimization* is that mathematical convergence proofs can be given. (For example, [8] proved convergence results for the algorithm implemented in [7].)

In Theorem 4 we show that *Legendre minimization* algorithms are equivalent to CCCP. The CCCP viewpoint emphasizes the geometry of the approach and complements the algebraic manipulations given in [8]. (Moreover, our results of the previous section show the generality of CCCP while, by contrast, the Legendre transform methods have been applied only on a case by case basis).

**Definition 1.** *Let $F(\vec{x})$ be a convex function. For each value $\vec{y}$ let $F^*(\vec{y}) = \min_{\vec{x}}\{F(\vec{x}) + \vec{y}\cdot\vec{x}.\}$. Then $F^*(\vec{y})$ is concave and is the Legendre transform of $F(\vec{x})$. Moreover, $F(\vec{x}) = \max_{\vec{y}}\{F^*(\vec{y}) - \vec{y}\cdot\vec{x}\}$.*

**Property 1.** *$F(.)$ and $F^*(.)$ are related by $\frac{\partial F^*}{\partial \vec{y}}(\vec{y}) = \{\frac{\partial F}{\partial \vec{x}}\}^{-1}(-\vec{y})$, $-\frac{\partial F}{\partial \vec{x}}(\vec{x}) = \{\frac{\partial F^*}{\partial \vec{y}}\}^{-1}(\vec{x})$. (By $\{\frac{\partial F^*}{\partial \vec{y}}\}^{-1}(\vec{x})$ we mean the value $\vec{y}$ such that $\frac{\partial F^*}{\partial \vec{y}}(\vec{y}) = \vec{x}$.)*

**Theorem 4.** *Let $E_1(\vec{x}) = f(\vec{x}) + g(\vec{x})$ and $E_2(\vec{x}, \vec{y}) = f(\vec{x}) + \vec{x}\cdot\vec{y} + h(\vec{y})$, where $f(.), h(.)$ are convex functions and $g(.)$ is concave. Then applying CCCP to $E_1(\vec{x})$ is equivalent to minimizing $E_2(\vec{x}, \vec{y})$ with respect to $\vec{x}$ and $\vec{y}$ alternatively (for suitable choices of $g(.)$ and $h(.)$.*

Proof. *We can write $E_1(\vec{x}) = f(\vec{x}) + \min_{\vec{y}}\{g^*(\vec{y}) + \vec{x}\cdot\vec{y}\}$ where $g^*(.)$ is the Legendre transform of $g(.)$ (identify $g(.)$ with $F^*(.)$ and $g^*(.)$ with $F(.)$ in definition 1). Thus minimizing $E_1(\vec{x})$ with respect to $\vec{x}$ is equivalent to minimizing $\hat{E}_1(\vec{x}, \vec{y}) = f(\vec{x}) + \vec{x}\cdot\vec{y} + g^*(\vec{y})$ with respect to $\vec{x}$ and $\vec{y}$. (Alternatively, we can set $g^*(\vec{y}) = h(\vec{y})$ in the expression for $E_2(\vec{x}, \vec{y})$ and obtain a cost function $\hat{E}_2(\vec{x}) = f(\vec{x}) + g(\vec{x})$.) Alternatively minimization over $\vec{x}$ and $\vec{y}$ gives: (i) $\partial f/\partial \vec{x} = \vec{y}$ to determine $\vec{x}_{t+1}$ in terms of $\vec{y}_t$, and (ii) $\partial g^*/\partial \vec{y} = \vec{x}$ to determine $\vec{y}_t$ in terms of $\vec{x}_t$ which, by Property 1 of the Legendre transform is equivalent to setting $\vec{y} = -\partial g/\partial \vec{x}$. Combining these two stages gives CCCP:*

$$\frac{\partial f}{\partial \vec{x}}(\vec{x}_{t+1}) = -\frac{\partial g}{\partial \vec{x}}(\vec{x}_t).$$

# 4 Examples of CCCP

We now illustrate CCCP by giving four examples: (i) discrete time dynamical systems for the mean field Potts model, (ii) an EM algorithm for the elastic net, (iii) a discrete (Sinkhorn) algorithm for solving the linear assignment problem, and (iv) the Generalized Iterative Scaling (GIS) algorithm for parameter estimation.

**Example 1.** *Discrete Time Dynamical Systems for the Mean Field Potts Model. These attempt to minimize discrete energy functions of form $E[V] = \sum_{i,j,a,b} \hat{T}_{ijab} V_{ia} V_{jb} + \sum_{ia} \theta_{ia} V_{ia}$, where the $\{V_{ia}\}$ take discrete values $\{0,1\}$ with linear constraints $\sum_i V_{ia} = 1, \ \forall a$.*

Discussion. *Mean field algorithms minimize a continuous effective energy $E_{eff}[S;T]$ to obtain a minimum of the discrete energy $E[V]$ in the limit as $T \mapsto 0$. The $\{S_{ia}\}$ are continuous variables in the range $[0,1]$ and correspond to (approximate) estimates of the mean states of the $\{V_{ia}\}$. As described in [12], to ensure that the minima of $E[V]$ and $E_{eff}[S;T]$ all coincide (as $T \mapsto 0$) it is sufficient that $\hat{T}_{ijab}$ be negative definite. Moreover, this can be attained by adding a term $-K\sum_{ia} V_{ia}^2$ to $E[V]$ (for sufficiently large $K$) without altering the structure of the minima of $E[V]$. Hence, without loss of generality we can consider $\sum_{i,j,a,b} \hat{T}_{ijab} V_{ia} V_{jb}$ to be a concave function.*

*We impose the linear constraints by adding a Lagrange multiplier term $\sum_a p_a\{\sum_i V_{ia} - 1\}$ to the energy where the $\{p_a\}$ are the Lagrange multipliers. The effective energy becomes:*

$$E_{eff}[S] = \sum_{i,j,a,b} \hat{T}_{ijab} S_{ia} S_{jb} + \sum_{ia} \theta_{ia} S_{ia} + T \sum_{ia} S_{ia} \log S_{ia} + \sum_a p_a\{\sum_i S_{ia} - 1\}. \quad (3)$$

*We can then incorporate the Lagrange multiplier term into the convex part. This gives: $E_{vex}[S] = T\sum_{ia} S_{ia} \log S_{ia} + \sum_a p_a\{\sum_i S_{ia} - 1\}$ and $E_{cave}[S] = \sum_{i,j,a,b} \hat{T}_{ijab} S_{ia} S_{jb} + \sum_{ia} \theta_{ia} S_{ia}$. Taking derivatives yields: $\frac{\partial}{\partial S_{ia}} E_{vex}[S] = T \log S_{ia} + p_a$ and $\frac{\partial}{\partial S_{ia}} E_{cave}[S] = 2\sum_{j,b} \hat{T}_{ijab} S_{jb} + \theta_{ia}$. Applying CCCP by setting $\frac{\partial E_{vex}}{\partial S_{ia}}(S^{t+1}) = -\frac{\partial E_{cave}}{\partial S_{ia}}(S^t)$ gives $T\{1 + \log S_{ia}(t+1)\} + p_a = -2\sum_{j,b} \hat{T}_{ijab} S_{jb}(t) - \theta_{ia}$. We solve for the Lagrange multipliers $\{p_a\}$ by imposing the constraints $\sum_i S_{ia}(t+1) = 1, \ \forall \ a$. This gives a discrete update rule:*

$$S_{ia}(t+1) = \frac{e^{(-1/T)\{2\sum_{j,b} \hat{T}_{ijab} S_{jb}(t) + \theta_{ia}\}}}{\sum_c e^{(-1/T)\{2\sum_{j,b} \hat{T}_{ijcb} S_{jb}(t) + \theta_{ic}\}}}. \quad (4)$$

*Algorithms of this type were derived in [10], [3] using different design principles.*

Our second example relates to the ubiquitous EM algorithm. In general EM and CCCP give different algorithms but in some cases they are identical. The EM algorithm seeks to estimate a variable $f^* = \arg\max_f \log \sum_{\{l\}} P(f, l)$, where $\{f\}, \{l\}$ are variables that depend on the specific problem formulation. It was shown in [4] that this is equivalent to *minimizing* the following effective energy with respect to the variables $f$ and $\hat{P}(l)$: $E_{eff}[f, \hat{P}(l)] = -\frac{1}{\beta} \sum_l \hat{P}(l) \log P(f, l) + \frac{1}{\beta} \sum_{\{l\}} \hat{P}(l) \log \hat{P}(l)$. To apply CCCP to an effective energy like this we need either: (a) to decompose $E_{eff}[f, \hat{P}(l)]$ into convex and concave functions of $f, \hat{P}(l)$, or (b) to eliminate either

variable and obtain a convex concave decomposition in the remaining variable (cf. Theorem 4). We illustrate (b) for the elastic net [2]. (See Yuille and Rangarajan, in preparation, for an illustration of (a)).

**Example 2.** *The elastic net attempts to solve the Travelling Salesman Problem (TSP) by finding the shortest tour through a set of cities at positions $\{\vec{x}_i\}$. The elastic net is represented by a set of nodes at positions $\{\vec{y}_a\}$ with variables $\{S_{ia}\}$ that determine the correspondence between the cities and the nodes of the net. Let $E_{eff}[S, \vec{y}]$ be the effective energy for the elastic net, then the $\{\vec{y}\}$ variables can be eliminated and the resulting $E_S[S]$ can be minimized using CCCP. (Note that the standard elastic net only enforces the second set of linear constraints).*

Discussion. *The elastic net energy function can be expressed as [11]:*

$$E_{eff}[S, \vec{y}] = \sum_{ia} S_{ia}(\vec{x}_i - \vec{y}_a)^2 + \gamma \sum_{a,b} \vec{y}_a A_{ab} \vec{y}_b + T \sum_{i,a} S_{ia} \log S_{ia}, \qquad (5)$$

*where we impose the conditions $\sum_a S_{ia} = 1$, $\forall\ i$ and $\sum_i S_{ia} = 1$, $\forall\ a$.*

*The EM algorithm can be applied to estimate the $\{\vec{y}_a\}$. Alternatively we can solve for the $\{\vec{y}_a\}$ variables to obtain $\vec{y}_b = \sum_{i,a} P_{ab} S_{ia} \vec{x}_i$ where $\{P_{ab}\} = \{\delta_{ab} + 2\gamma A_{ab}\}^{-1}$. We substitute this back into $E_{eff}[S, \vec{y}]$ to get a new energy $E_S[S]$ given by:*

$$E_S[S] = - \sum_{i,j,a,b} S_{ia} S_{jb} \{P_{ba} \vec{x}_i \cdot \vec{x}_j\} + T \sum_{i,a} S_{ia} \log S_{ia}. \qquad (6)$$

*Once again this is a sum of a concave and a convex part (the first term is concave because of the minus sign and the fact that $\{P_{ba}\}$ and $\vec{x}_i \cdot \vec{x}_j$ are both positive semi-definite.) We can now apply CCCP and obtain the standard EM algorithm for this problem. (See Yuille and Rangarajan, in preparation, for more details).*

Our final example is a discrete iterative algorithm to solve the linear assignment problem. This algorithm was reported by Kosowsky and Yuille in [5] where it was also shown to correspond to the well-known Sinkhorn algorithm [9]. We now show that both Kosowsky and Yuille's linear assignment algorithm, and hence Sinkhorn's algorithm are examples of CCCP (after a change of variables).

**Example 3.** *The linear assignment problem seeks to find the permutation matrix $\{\prod_{ia}\}$ which minimizes the energy $E[\prod] = \sum_{ia} \prod_{ia} A_{ia}$, where $\{A_{ia}\}$ is a set of assignment values. As shown in [5] this is equivalent to minimizing the (convex) $E_P[P]$ energy given by $E_P[P] = \sum_a p_a + \frac{1}{\beta} \sum_i \log \sum_a e^{-\beta(A_{ia}+p_a)}$, where the solution is given by $\prod_{ia}^* = e^{-\beta(A_{ia}+p_a)} / \sum_b e^{-\beta(A_{ib}+p_b)}$ rounded off to the nearest integer (for sufficiently large $\beta$). The iterative algorithm to minimize $E_P[P]$ (which can be re-expressed as Sinkhorn's algorithm, see [5]) is of form:*

$$e^{\beta p_a^{t+1}} = \sum_i \frac{e^{-\beta A_{ia}}}{\sum_b e^{-\beta(A_{ib}+p_b^t)}}, \qquad (7)$$

*and can be re-expressed as CCCP.*

Discussion. *By performing the change of coordinates $\beta p_a = -\log r_a\ \forall\ a$ (for $r_a >$*

$0, \forall a$) we can re-express the $E_P[P]$ energy as:

$$E_r[r] = -\frac{1}{\beta} \sum_a \log r_a + \frac{1}{\beta} \sum_i \log \sum_a r_a e^{-\beta A_{ia}}. \qquad (8)$$

Observe that the first term of $E_r[r]$ is convex and the second term is concave (this can be verified by calculating the Hessian). Applying CCCP gives the update rule:

$$\frac{1}{r_a^{t+1}} = \sum_i \frac{e^{-\beta A_{ia}}}{\sum_b e^{-\beta A_{ib}} r_b^t}, \qquad (9)$$

which corresponds to equation (7).

**Example 4.** *The Generalized Iterative Scaling (GIS) Algorithm [1] for estimating parameters in parallel.*

Discussion. *The GIS algorithm is designed to estimate the parameter $\vec{\lambda}$ of a distribution $P(\vec{x} : \vec{\lambda}) = e^{\vec{\lambda} \cdot \vec{\phi}(\vec{x})}/Z[\vec{\lambda}]$ so that $\sum_{\vec{x}} P(\vec{x}; \vec{\lambda}) \vec{\phi}(\vec{x}) = \vec{h}$, where $\vec{h}$ are observation data (with components indexed by $\mu$). It is assumed that $\phi_\mu(\vec{x}) \geq 0$, $\forall \mu, \vec{x}$, $h_\mu \geq 0$, $\forall \mu$, and $\sum_\mu \phi_\mu(\vec{x}) = 1$, $\forall \vec{x}$ and $\sum_\mu h_\mu = 1$. (All estimation problems of this type can be transformed into this form [1]).*

*Darroch and Ratcliff [1] prove that the following GIS algorithm is guaranteed to converge to value $\vec{\lambda}^*$ that minimizes the (convex) cost function $E(\vec{\lambda}) = \log Z[\vec{\lambda}] - \vec{\lambda} \cdot \vec{h}$ and hence satisfies $\sum_{\vec{x}} P(\vec{x}; \vec{\lambda}^*) \vec{\phi}(\vec{x}) = \vec{h}$. The GIS algorithms is given by:*

$$\vec{\lambda}^{t+1} = \vec{\lambda}^t - \log \vec{h}^t + \log \vec{h}, \qquad (10)$$

*where $\vec{h}^t = \sum_{\vec{x}} P(\vec{x}; \vec{\lambda}^t) \vec{\phi}(\vec{x})$ (evaluate $\log \vec{h}$ componentwise: $(\log \vec{h})_\mu = \log h_\mu$.)*

*To show that GIS can be reformulated as CCCP, we introduce a new variable $\vec{\beta} = e^{\vec{\lambda}}$ (componentwise). We reformulate the problem in terms of minimizing the cost function $E_\beta[\vec{\beta}] = \log Z[\log(\vec{\beta})] - \vec{h} \cdot (\log \vec{\beta})$. A straightforward calculation shows that $-\vec{h} \cdot (\log \vec{\beta})$ is a convex function of $\vec{\beta}$ with first derivative being $-\vec{h}/\vec{\beta}$ (where the division is componentwise). The first derivative of $\log Z[\log(\vec{\beta})]$ is $(1/\vec{\beta}) \sum_{\vec{x}} \vec{\phi}(\vec{x}) P(\vec{x} : \log \beta)$ (evaluated componentwise). To show that $\log Z[\log(\vec{\beta})]$ is concave requires computing its Hessian and applying the Cauchy-Schwarz inequality using the fact that $\sum_\mu \phi_\mu(\vec{x}) = 1$, $\forall \vec{x}$ and that $\phi_\mu(\vec{x}) \geq 0$, $\forall \mu, \vec{x}$. We can therefore apply CCCP to $E_\beta[\vec{\beta}]$ which yields $1/\vec{\beta}^{t+1} = 1/\vec{\beta}^t \times 1/\vec{h} \times \vec{h}^t$ (componentwise), which is GIS (by taking logs and using $\log \beta = \vec{\lambda}$).*

# 5 Conclusion

CCCP is a general principle which can be used to construct discrete time iterative dynamical systems for almost any energy minimization problem. It gives a geometric perspective on *Legendre minimization* (though not on *Legendre min-max*).

We have illustrated that several existing discrete time iterative algorithms can be re-interpreted in terms of CCCP (see Yuille and Rangarajan, in preparation, for other

examples). Therefore CCCP gives a novel way of thinking about and classifying existing algorithms. Moreover, CCCP can also be used to construct novel algorithms. See, for example, recent work [13] where CCCP was used to construct a double loop algorithm to minimize the Bethe/Kikuchi free energy (which are generalizations of the mean field free energy).

There are interesting connections between our results and those known to mathematicians. After this work was completed we found that a result similar to Theorem 2 had appeared in an unpublished technical report by D. Geman. There also are similarities to the work of Hoang Tuy who has shown that any arbitrary closed set is the projection of a difference of two convex sets in a space with one more dimension. (See http://www.mai.liu.se/Opt/MPS/News/tuy.html).

## Acknowledgements

We thank James Coughlan and Yair Weiss for helpful conversations. Max Welling gave useful feedback on this manuscript. We thank the National Institute of Health (NEI) for grant number RO1-EY 12691-01.

## References

[1] J.N. Darroch and D. Ratcliff. "Generalized Iterative Scaling for Log-Linear Models". *The Annals of Mathematical Statistics*. Vol. 43. No. 5, pp 1470-1480. 1972.

[2] R. Durbin, R. Szeliski and A.L. Yuille." An Analysis of an Elastic net Approach to the Traveling Salesman Problem". *Neural Computation*. **1**, pp 348-358. 1989.

[3] I.M. Elfadel "Convex potentials and their conjugates in analog mean-field optimization". *Neural Computation*. Volume 7. Number 5. pp. 1079-1104. 1995.

[4] R. Hathaway. "Another Interpretation of the EM Algorithm for Mixture Distributions". *Statistics and Probability Letters*. Vol. 4, pp 53-56. 1986.

[5] J. Kosowsky and A.L. Yuille. "The Invisible Hand Algorithm: Solving the Assignment Problem with Statistical Physics". *Neural Networks.*, Vol. 7, No. 3, pp 477-490. 1994.

[6] E. Mjolsness and C. Garrett. "Algebraic Transformations of Objective Functions". *Neural Networks*. Vol. 3, pp 651-669.

[7] A. Rangarajan, S. Gold, and E. Mjolsness. "A Novel Optimizing Network Architecture with Applications". *Neural Computation*, 8(5), pp 1041-1060. 1996.

[8] A. Rangarajan, A.L. Yuille, S. Gold. and E. Mjolsness."A Convergence Proof for the Softassign Quadratic assignment Problem". In *Proceedings of NIPS'96*. Denver. Colorado. 1996.

[9] R. Sinkhorn. "A Relationship Between Arbitrary Positive Matrices and Doubly Stochastic Matrices". *Ann. Math. Statist.*. 35, pp 876-879. 1964.

[10] F.R. Waugh and R.M. Westervelt. "Analog neural networks with local competition: I. Dynamics and stability". *Physical Review E*, 47(6), pp 4524-4536. 1993.

[11] A.L. Yuille. "Generalized Deformable Models, Statistical Physics and Matching Problems," *Neural Computation*, **2** pp 1-24. 1990.

[12] A.L. Yuille and J.J. Kosowsky. "Statistical Physics Algorithms that Converge." Neural Computation. **6**, pp 341-356. 1994.

[13] A.L. Yuille. "A Double-Loop Algorithm to Minimize the Bethe and Kikuchi Free Energies". Neural Computation. In press. 2002.
